# A Matching Pursuit Approach to Sparse Gaussian Process Regression

**S. Sathiya Keerthi**
Yahoo! Research Labs
210 S. DeLacey Avenue
Pasadena, CA 91105
selvarak@yahoo-inc.com

**Wei Chu**
Gatsby Computational Neuroscience Unit
University College London
London, WC1N 3AR, UK
chuwei@gatsby.ucl.ac.uk

## Abstract

In this paper we propose a new basis selection criterion for building sparse GP regression models that provides promising gains in accuracy as well as efficiency over previous methods. Our algorithm is much faster than that of Smola and Bartlett, while, in generalization it greatly outperforms the information gain approach proposed by Seeger et al, especially on the quality of predictive distributions.

## 1 Introduction

Bayesian Gaussian processes provide a promising probabilistic kernel approach to supervised learning tasks. The advantage of Gaussian process (GP) models over non-Bayesian kernel methods, such as support vector machines, comes from the explicit probabilistic formulation that yields predictive distributions for test instances and allows standard Bayesian techniques for model selection. The cost of training GP models is $\mathcal{O}(n^3)$ where $n$ is the number of training instances, which results in a huge computational cost for large data sets. Furthermore, when predicting a test case, a GP model requires $\mathcal{O}(n)$ cost for computing the mean and $\mathcal{O}(n^2)$ cost for computing the variance. These heavy scaling properties obstruct the use of GPs in large scale problems.

Recently, sparse GP models which bring down the complexity of training as well as testing have attracted considerable attention. Williams and Seeger (2001) applied the Nyström method to calculate a reduced rank approximation of the original $n \times n$ kernel matrix. Csató and Opper (2002) developed an on-line algorithm to maintain a sparse representation of the GP models. Smola and Bartlett (2001) proposed a forward selection scheme to approximate the log posterior probability. Candela (2004) suggested a promising alternative criterion by maximizing the approximate model evidence. Seeger et al. (2003) presented a very fast greedy selection method for building sparse GP regression models. All of these methods make efforts to select an informative subset of the training instances for the predictive model. This subset is usually referred to as the set of *basis vectors*, denoted as $\mathcal{I}$. The maximal size of $\mathcal{I}$ is usually limited by a value $d_{\max}$. As $d_{\max} \ll n$, the sparseness greatly alleviates the computational burden in both training and prediction of the GP models. The performance of the resulting sparse GP models crucially depends on the criterion used in the basis vector selection. Motivated by the ideas of Matching Pursuit (Vincent and Bengio, 2002), we propose a new criterion of greedy forward selection for sparse GP models.

Our algorithm is closely related to that of Smola and Bartlett (2001), but the criterion we propose is much more efficient. Compared with the information gain method of Seeger et al. (2003) our approach yields clearly better generalization performance, while essentially having the same algorithm complexity. We focus only on regression in this paper, but the main ideas are applicable to other supervised learning tasks.

The paper is organized as follows: in Section 2 we present the probabilistic framework for sparse GP models; in Section 3 we describe our method of greedy forward selection after motivating it via the previous methods; in Section 4 we discuss some issues in model adaptation; in Section 5 we report results of numerical experiments that demonstrate the effectiveness of our new method.

## 2   Sparse GPs for regression

In regression problems, we are given a training data set composed of $n$ samples. Each sample is a pair of an input vector $\boldsymbol{x}_i \in \mathbb{R}^m$ and its corresponding target $y_i \in \mathbb{R}$. The true function value at $\boldsymbol{x}_i$ is represented as an unobservable latent variable $f(\boldsymbol{x}_i)$ and the target $y_i$ is a noisy measurement of $f(\boldsymbol{x}_i)$. The goal is to construct a predictive model that estimates the relationship $\boldsymbol{x} \mapsto f(\boldsymbol{x})$.

**Gaussian process regression.** In standard GPs for regression, the latent variables $\{f(\boldsymbol{x}_i)\}$ are random variables in a zero mean Gaussian process indexed by $\{\boldsymbol{x}_i\}$. The prior distribution of $\{f(\boldsymbol{x}_i)\}$ is a multivariate joint Gaussian, denoted as $\mathcal{P}(\boldsymbol{f}) = \mathcal{N}(\boldsymbol{f}; 0, \mathbf{K})$, where $\boldsymbol{f} = [f(\boldsymbol{x}_1), \ldots, f(\boldsymbol{x}_n)]^T$ and $\mathbf{K}$ is the $n \times n$ covariance matrix whose $ij$-th element is $\mathcal{K}(\boldsymbol{x}_i, \boldsymbol{x}_j)$, $\mathcal{K}$ being the kernel function. The likelihood is essentially a model of the measurement noise, which is usually evaluated as a product of independent Gaussian noises, $\mathcal{P}(\boldsymbol{y}|\boldsymbol{f}) = \mathcal{N}(\boldsymbol{y}; \boldsymbol{f}, \sigma^2 \mathbf{I})$, where $\boldsymbol{y} = [y_1, \ldots, y_n]^T$ and $\sigma^2$ is the noise variance. The posterior distribution $\mathcal{P}(\boldsymbol{f}|\boldsymbol{y}) \propto \mathcal{P}(\boldsymbol{y}|\boldsymbol{f})\mathcal{P}(\boldsymbol{f})$ is also exactly a Gaussian:

$$\mathcal{P}(\boldsymbol{f}|\boldsymbol{y}) = \mathcal{N}(\boldsymbol{f}; \mathbf{K}\boldsymbol{\alpha}^\star, \sigma^2 \mathbf{K}(\mathbf{K} + \sigma^2 \mathbf{I})^{-1}) \tag{1}$$

where $\boldsymbol{\alpha}^\star = (\mathbf{K} + \sigma^2 \mathbf{I})^{-1}\boldsymbol{y}$. For any test instance $\boldsymbol{x}$, the predictive distribution is $\mathcal{N}(f(\boldsymbol{x}); \mu_{\boldsymbol{x}}, \sigma_{\boldsymbol{x}}^2)$ where $\mu_{\boldsymbol{x}} = \boldsymbol{k}^T(\mathbf{K} + \sigma^2 \mathbf{I})^{-1}\boldsymbol{y} = \boldsymbol{k}^T \boldsymbol{\alpha}^\star$, $\sigma_{\boldsymbol{x}}^2 = \mathcal{K}(\boldsymbol{x}, \boldsymbol{x}) - \boldsymbol{k}^T(\mathbf{K} + \sigma^2 \mathbf{I})^{-1}\boldsymbol{k}$, and $\boldsymbol{k} = [\mathcal{K}(\boldsymbol{x}_1, \boldsymbol{x}), \ldots, \mathcal{K}(\boldsymbol{x}_n, \boldsymbol{x})]^T$. The computational cost of training is $\mathcal{O}(n^3)$, which mainly comes from the need to invert the matrix $(\mathbf{K} + \sigma^2 \mathbf{I})$ and obtain the vector $\boldsymbol{\alpha}^\star$. For doing predictions of a test instance the cost is $\mathcal{O}(n)$ to compute the mean and $\mathcal{O}(n^2)$ for computing the variance. This heavy scaling with respect to $n$ makes the use of standard GP computationally prohibitive on large datasets.

**Projected latent variables.** Seeger et al. (2003) gave a neat method for working with a reduced number of latent variables, laying the foundation for forming sparse GP models. In this section we review their ideas. Instead of assuming $n$ latent variables for all the training instances, sparse GP models assume only $d$ latent variables placed at some chosen basis vectors $\{\tilde{\boldsymbol{x}}_i\}$, denoted as a column vector $\boldsymbol{f}_{\mathcal{I}} = [f(\tilde{\boldsymbol{x}}_1), \ldots, f(\tilde{\boldsymbol{x}}_d)]^T$. The prior distribution of the sparse GP is a joint Gaussian over $\boldsymbol{f}_{\mathcal{I}}$ only, i.e.,

$$\mathcal{P}(\boldsymbol{f}_{\mathcal{I}}) = \mathcal{N}(\boldsymbol{f}_{\mathcal{I}}; 0, \mathbf{K}_{\mathcal{I}}) \tag{2}$$

where $\mathbf{K}_{\mathcal{I}}$ is the $d \times d$ covariance matrix of the basis vectors whose $ij$-th element is $\mathcal{K}(\tilde{\boldsymbol{x}}_i, \tilde{\boldsymbol{x}}_j)$.

These latent variables are then projected to all the training instances. Under the imposed joint Gaussian prior, the conditional mean at the training instances is $\mathbf{K}_{\mathcal{I},\cdot}^T \mathbf{K}_{\mathcal{I}}^{-1} \boldsymbol{f}_{\mathcal{I}}$, where $\mathbf{K}_{\mathcal{I},\cdot}$ is a $d \times n$ matrix of the covariance functions between the basis vectors and all the training instances. The likelihood can be evaluated by these projected latent variables as follows

$$\mathcal{P}(\boldsymbol{y}|\boldsymbol{f}_{\mathcal{I}}) = \mathcal{N}(\boldsymbol{y}; \mathbf{K}_{\mathcal{I},\cdot}^T \mathbf{K}_{\mathcal{I}}^{-1} \boldsymbol{f}_{\mathcal{I}}, \sigma^2 \mathbf{I}) \tag{3}$$

The posterior is $\mathcal{P}(\boldsymbol{f}_{\mathcal{I}}|\boldsymbol{y}) = \mathcal{N}(\boldsymbol{f}_{\mathcal{I}}; \mathbf{K}_{\mathcal{I}} \boldsymbol{\alpha}_{\mathcal{I}}^{\star}, \sigma^2 \mathbf{K}_{\mathcal{I}}(\sigma^2 \mathbf{K}_{\mathcal{I}} + \mathbf{K}_{\mathcal{I},\cdot} \mathbf{K}_{\mathcal{I},\cdot}^T)^{-1} \mathbf{K}_{\mathcal{I}})$, where $\boldsymbol{\alpha}_{\mathcal{I}}^{\star} = (\sigma^2 \mathbf{K}_{\mathcal{I}} + \mathbf{K}_{\mathcal{I},\cdot} \mathbf{K}_{\mathcal{I},\cdot}^T)^{-1} \mathbf{K}_{\mathcal{I},\cdot}\, \boldsymbol{y}$. The predictive distribution at any test instance $\boldsymbol{x}$ is $\mathcal{N}(f(\boldsymbol{x}); \tilde{\mu}_{\boldsymbol{x}}, \tilde{\sigma}_{\boldsymbol{x}}^2)$, where $\tilde{\mu}_{\boldsymbol{x}} = \tilde{\boldsymbol{k}}^T \boldsymbol{\alpha}_{\mathcal{I}}^{\star}$, $\tilde{\sigma}_{\boldsymbol{x}}^2 = \mathcal{K}(\boldsymbol{x}, \boldsymbol{x}) - \tilde{\boldsymbol{k}}^T \mathbf{K}_{\mathcal{I}}^{-1} \tilde{\boldsymbol{k}}^T + \sigma^2 \tilde{\boldsymbol{k}}^T (\sigma^2 \mathbf{K}_{\mathcal{I}} + \mathbf{K}_{\mathcal{I},\cdot} \mathbf{K}_{\mathcal{I},\cdot}^T)^{-1} \tilde{\boldsymbol{k}}$, and $\tilde{\boldsymbol{k}}$ is a column vector of the covariance functions between the basis vectors and the test instance $\boldsymbol{x}$, i.e. $\tilde{\boldsymbol{k}} = [\mathcal{K}(\tilde{\boldsymbol{x}}_1, \boldsymbol{x}), \ldots, \mathcal{K}(\tilde{\boldsymbol{x}}_d, \boldsymbol{x})]^T$.

While the cost of training the full GP model is $\mathcal{O}(n^3)$, the training complexity of sparse GP models is only $\mathcal{O}(n d_{\max}^2)$. This corresponds to the cost of forming $\mathbf{K}_{\mathcal{I}}^{-1}$, $(\sigma^2 \mathbf{K}_{\mathcal{I}} + \mathbf{K}_{\mathcal{I},\cdot} \mathbf{K}_{\mathcal{I},\cdot}^T)^{-1}$ and $\boldsymbol{\alpha}_{\mathcal{I}}^{\star}$. Thus, if $d_{\max}$ is not big, learning on large datasets is feasible via sparse GP models. Also, for these sparse models, prediction for each test instance costs $\mathcal{O}(d_{\max})$ for the mean and $\mathcal{O}(d_{\max}^2)$ for the variance.

Generally the basis vectors can be placed anywhere in the input space $\mathbb{R}^m$. Since training instances usually cover the input space of interest quite well, it is quite reasonable to select basis vectors from just the set of training instances. For a given problem $d_{\max}$ is chosen to be as large as possible subject to constraints on computational time in training and/or testing. Then we use some basis selection method to find $\mathcal{I}$ of size $d_{\max}$. This important step is taken up in section 3.

**A Useful optimization formulation.** As pointed out by Smola and Bartlett (2001), it is useful to view the determination of the mean of the posterior as coming from an optimization problem. This viewpoint helps in the selection of basis vectors. The mean of the posterior distribution is exactly the maximum a posteriori (MAP) estimate, and it is possible to give an equivalent parametric representation of the latent variables as $\boldsymbol{f} = \mathbf{K}\boldsymbol{\alpha}$, where $\boldsymbol{\alpha} = [\alpha_1, \ldots, \alpha_n]^T$. The MAP estimate of the full GP is equivalent to minimizing the negative logarithm of the posterior (1):

$$\min_{\boldsymbol{\alpha}} \pi(\boldsymbol{\alpha}) := \frac{1}{2} \boldsymbol{\alpha}^T (\sigma^2 \mathbf{K} + \mathbf{K}^T \mathbf{K}) \boldsymbol{\alpha} - \boldsymbol{y}^T \mathbf{K} \boldsymbol{\alpha} \qquad (4)$$

Similarly, using $\boldsymbol{f}_{\mathcal{I}} = \mathbf{K}_{\mathcal{I}} \boldsymbol{\alpha}_{\mathcal{I}}$ for sparse GP models, the MAP estimate of the sparse GP is equivalent to minimizing the negative logarithm of the posterior, $\mathcal{P}(\boldsymbol{f}_{\mathcal{I}}|\boldsymbol{y})$:

$$\min_{\boldsymbol{\alpha}_{\mathcal{I}}} \tilde{\pi}(\boldsymbol{\alpha}_{\mathcal{I}}) := \frac{1}{2} \boldsymbol{\alpha}_{\mathcal{I}}^T (\sigma^2 \mathbf{K}_{\mathcal{I}} + \mathbf{K}_{\mathcal{I},\cdot} \mathbf{K}_{\mathcal{I},\cdot}^T) \boldsymbol{\alpha}_{\mathcal{I}} - \boldsymbol{y}^T \mathbf{K}_{\mathcal{I},\cdot}^T \boldsymbol{\alpha}_{\mathcal{I}} \qquad (5)$$

Suppose $\boldsymbol{\alpha}$ in (4) is composed of two parts, $\boldsymbol{\alpha} = [\boldsymbol{\alpha}_{\mathcal{I}} ; \boldsymbol{\alpha}_{\mathcal{R}}]$ where $\mathcal{I}$ denotes the set of basis vectors and $\mathcal{R}$ denotes the remaining instances. Interestingly, as pointed out by Seeger et al. (2003), the optimization problem (5) is same as minimizing $\pi(\boldsymbol{\alpha})$ in (4) using $\boldsymbol{\alpha}_{\mathcal{I}}$ only, i.e., with the constraint, $\boldsymbol{\alpha}_{\mathcal{R}} = 0$. In other words, the basis vectors of the sparse GPs can be selected to minimize the negative log-posterior of the full GPs, $\pi(\boldsymbol{\alpha})$ defined as in (4).

## 3  Selection of basis functions

The most crucial element of the sparse GP approach of the previous section is the choice of $\mathcal{I}$, the set of basis vectors, which we take to be a subset of the training vectors. The cheapest method is to select the basis vectors at *random* from the training data set. But, such a choice will not work well when $d_{\max}$ is much smaller than $n$. A principled approach is to select $\mathcal{I}$ that makes the corresponding sparse GP approximate well, the posterior distribution of the full GP. The optimization formulation of the previous section is useful here. It would be ideal to choose, among all subsets, $\mathcal{I}$ of size $d_{\max}$, the one that gives the best value of $\tilde{\pi}$ in (5). But, this requires a combinatorial search that is infeasible for large problems. A practical approach is to do greedy forward selection. This is the approach used in previous methods as well as in our method of this paper.

Before we go into the details of the methods, let us give a brief discussion of the time complexities associated with forward selection. There are two costs involved. (1) There is a

basic cost associated with updating of the sparse GP solution, given a sequence of chosen basis functions. Let us refer to this cost as $T_{\text{basic}}$. This cost is the same for all forward selection methods, and is $\mathcal{O}(nd_{\max}^2)$. (2) Then, depending on the basis selection method, there is the cost associated with basis selection. We will refer to the accumulated value of this cost for choosing all $d_{\max}$ basis functions as $T_{\text{selection}}$. Forward basis selection methods differ in the way they choose effective basis functions while keeping $T_{\text{selection}}$ small. It is useful to note that the total cost associated with the *random* basis selection method mentioned earlier is just $T_{\text{basic}} = \mathcal{O}(nd_{\max}^2)$. This cost forms a baseline for comparison.

**Smola and Bartlett's method.** Consider the typical situation in forward selection where we have a current working set $\mathcal{I}$ and we are interested in choosing the next basis vector, $x_i$. The method of Smola and Bartlett (2001) evaluates each given $x_i \notin \mathcal{I}$ by trying its complete inclusion, i.e., set $\mathcal{I}' = \mathcal{I} \cup \{\boldsymbol{x}_i\}$ and optimize $\pi(\boldsymbol{\alpha})$ using $\boldsymbol{\alpha}_{\mathcal{I}'} = [\boldsymbol{\alpha}_{\mathcal{I}} ; \alpha_i]$. Thus, their selection criterion for the instance $\boldsymbol{x}_i \notin \mathcal{I}$ is the decrease in $\pi(\boldsymbol{\alpha})$ that can be obtained by allowing both $\boldsymbol{\alpha}_{\mathcal{I}}$ and $\alpha_i$ as variables to be non-zero. The minimal value of $\pi(\boldsymbol{\alpha})$ can be obtained by solving $\min_{\boldsymbol{\alpha}_{\mathcal{I}'}} \tilde{\pi}(\boldsymbol{\alpha}_{\mathcal{I}'})$ defined in (5). This costs $\mathcal{O}(nd)$ time for each candidate, $\boldsymbol{x}_i$, where $d$ is the size of the current set, $\mathcal{I}$. If all $\boldsymbol{x}_i \notin \mathcal{I}$ need to be tried, it will lead to $\mathcal{O}(n^2 d)$ cost. Accumulated till $d_{\max}$ basis functions are added, this leads to a $T_{\text{selection}}$ that has $\mathcal{O}(n^2 d_{\max}^2)$ complexity, which is disproportionately higher than $T_{\text{basic}}$. Therefore, Smola and Bartlett (2001) resorted to a randomized scheme by considering only $\kappa$ basis elements randomly chosen from outside $\mathcal{I}$ during one basis selection. They used a value of $\kappa = 59$. For this randomized method, the complexity of $T_{\text{selection}}$ is $\mathcal{O}(\kappa nd_{\max}^2)$. Although, from a complexity viewpoint, $T_{\text{basic}}$ and $T_{\text{selection}}$ are same, it should be noted that the overall cost of the method is about 60 times that of $T_{\text{basic}}$.

**Seeger et al's information gain method.** Seeger et al. (2003) proposed a novel and very cheap heuristic criterion for basis selection. The "informativeness" of an input vector $\boldsymbol{x}_i \notin \mathcal{I}$ is scored by the information gain between the true posterior distribution, $\mathcal{P}(\boldsymbol{f}_{\mathcal{I}'}|\boldsymbol{y})$ and a posterior approximation, $\mathcal{Q}(\boldsymbol{f}_{\mathcal{I}'}|\boldsymbol{y})$, where $\mathcal{I}'$ denotes the new set of basis vectors after including a new element $\boldsymbol{x}_i$ into the current set $\mathcal{I}$. The posterior approximation $\mathcal{Q}(\boldsymbol{f}_{\mathcal{I}'}|\boldsymbol{y})$ ignores the dependencies between the latent variable $f(x_i)$ and the targets other than $y_i$. Due to this simplification, this value of information gain is computed in $\mathcal{O}(1)$ time, given the current predictive model represented by $\mathcal{I}$. Thus, the scores of all instances outside $\mathcal{I}$ can be efficiently evaluated in $\mathcal{O}(n)$ time, which makes this algorithm almost as fast as using random selection! The potential weakness of this algorithm might be the non-use of the correlation in the remaining instances $\{\boldsymbol{x}_i : \boldsymbol{x}_i \notin \mathcal{I}\}$.

**Post-backfitting approach.** The two methods presented above are extremes in efficiency: in Smola and Bartlett's method $T_{\text{selection}}$ is disproportionately larger than $T_{\text{basic}}$ while, in Seeger et al's method $T_{\text{selection}}$ is very much smaller than $T_{\text{basic}}$. In this section we introduce a moderate method that is effective and whose complexity is in between the two earlier methods. Our method borrows an idea from kernel matching pursuit.

Kernel Matching Pursuit (Vincent and Bengio, 2002) is a sparse method for ordinary least squares that consists of two general greedy sparse approximation schemes, called *pre-backfitting* and *post-backfitting*. It is worth pointing out that the same methods were also considered much earlier in Adler et al. (1996). Both methods can be generalized to select the basis vectors for sparse GPs. The *pre-backfitting* approach is very similar in spirit to Smola and Bartlett's method. Our method is an efficient selection criterion that is based on the *post-backfitting* idea. Recall that, given the current $\mathcal{I}$, the minimal value of $\pi(\boldsymbol{\alpha})$ when it is optimized using only $\boldsymbol{\alpha}_{\mathcal{I}}$ as variables is equivalent to $\min_{\boldsymbol{\alpha}_{\mathcal{I}}} \tilde{\pi}(\boldsymbol{\alpha}_{\mathcal{I}})$ as in (5). The minimizer, denoted as $\boldsymbol{\alpha}_{\mathcal{I}}^{\star}$, is given by

$$\boldsymbol{\alpha}_{\mathcal{I}}^{\star} = (\sigma^2 \mathbf{K}_{\mathcal{I}} + \mathbf{K}_{\mathcal{I},\cdot} \, \mathbf{K}_{\mathcal{I},\cdot}^T)^{-1} \mathbf{K}_{\mathcal{I},\cdot} \, \boldsymbol{y} \qquad (6)$$

Our scoring criterion for an instance $\boldsymbol{x}_i \notin \mathcal{I}$ is based on optimizing $\pi(\boldsymbol{\alpha})$ by fixing $\boldsymbol{\alpha}_{\mathcal{I}} = \boldsymbol{\alpha}_{\mathcal{I}}^{\star}$ and changing $\alpha_i$ only. The one-dimensional minimizer can be easily found as

$$\alpha_i^* = \frac{\mathbf{K}_{i,\cdot}^T (\boldsymbol{y} - \mathbf{K}_{\mathcal{I},\cdot}^T \boldsymbol{\alpha}_{\mathcal{I}}^\star) - \sigma^2 \tilde{\boldsymbol{k}}_i^T \boldsymbol{\alpha}_{\mathcal{I}}^\star}{\sigma^2 \mathcal{K}(\boldsymbol{x}_i, \boldsymbol{x}_i) + \mathbf{K}_{i,\cdot}^T \mathbf{K}_{i,\cdot}} \tag{7}$$

where $\mathbf{K}_{i,\cdot}$ is the $n \times 1$ matrix of covariance functions between $\boldsymbol{x}_i$ and all the training data, and $\tilde{\boldsymbol{k}}_i$ is a $d$ dimensional vector having $\mathcal{K}(\boldsymbol{x}_j, \boldsymbol{x}_i)$, $\boldsymbol{x}_j \in \mathcal{I}$. The selection score of the instance $\boldsymbol{x}_i$ is the decrease in $\pi(\boldsymbol{\alpha})$ achieved by the one dimensional optimization of $\alpha_i$, which can be written in closed form as

$$\Delta_i = \frac{1}{2} (\alpha_i^*)^2 \left( \sigma^2 \mathcal{K}(\boldsymbol{x}_i, \boldsymbol{x}_i) + \mathbf{K}_{i,\cdot}^T \mathbf{K}_{i,\cdot} \right) \tag{8}$$

where $\alpha_i^*$ is defined as in (7). Note that a full kernel column $\mathbf{K}_{i,\cdot}$ is required and so it costs $\mathcal{O}(n)$ time to compute (8). In contrast, for scoring one instance, Smola and Bartlett's method requires $\mathcal{O}(nd)$ time and Seeger et al's method requires $\mathcal{O}(1)$ time.

Ideally we would like to run over all $\boldsymbol{x}_i \notin \mathcal{I}$ and choose the instance which gives the largest decrease. This will need $\mathcal{O}(n^2)$ effort. Summing the cost till $d_{\max}$ basis vectors are selected, we get an overall complexity of $\mathcal{O}(n^2 d_{\max})$, which is much higher than $T_{\text{basic}}$. To restrict the overall complexity of $T_{\text{selection}}$ to $\mathcal{O}(nd_{\max}^2)$, we resort to a randomization scheme that selects a relatively good one rather than the best. Since it costs only $\mathcal{O}(n)$ time to evaluate our selection criterion in (8) for one instance, we can choose the next basis vector from a set of $d_{\max}$ instances randomly selected from outside of $\mathcal{I}$. Such a scheme keeps the overall complexity of $T_{\text{selection}}$ to $\mathcal{O}(nd_{\max}^2)$. But, from a practical point of view the scheme is expensive because the selection criterion (8) requires computing a full kernel row $\mathbf{K}_{i,\cdot}$ for each instance to be evaluated. As kernel evaluations could be very expensive, we propose a modified scheme to keep the number of such evaluations small.

Let us maintain a matrix cache, $\mathcal{C}$ of size $c \times n$, that contains $c$ rows of the full kernel matrix $\mathbf{K}$. At the beginning of the algorithm (when $\mathcal{I}$ is empty) we initialize $\mathcal{C}$ by randomly choosing $c$ training instances, computing the full kernel row, $\mathbf{K}_{i,\cdot}$ for the chosen $i$'s and putting them in the rows of $\mathcal{C}$. Each step corresponding to a new basis vector selection proceeds as follows. First we compute $\Delta_i$ for the $c$ instances corresponding to the rows of $\mathcal{C}$ and select the instance with the highest score for inclusion in $\mathcal{I}$. Let $x_j$ denote the chosen basis vector. Then we sort the remaining instances (that define $\mathcal{C}$) according to their $\Delta_i$ values. Finally, we randomly select $\kappa$ fresh instances (from outside of $\mathcal{I}$ and the vectors that define $\mathcal{C}$) to replace $x_j$ and the $\kappa - 1$ cached instances with the lowest score. Thus, in each basis selection step, we compute the criterion scores for $c$ instances, but evaluate full kernel rows only for $\kappa$ fresh instances. An important advantage of the above scheme is that, those basis elements which have very good scores, but are overtaken by another better element in a particular step, continue to remain in $\mathcal{C}$ and probably get to be selected in future basis selection steps. Like in Smola and Bartlett's method we use $\kappa = 59$. The value of $c$ can be set to be any integer between $\kappa$ and $d_{\max}$. For any $c$ in this range, the complexity of $T_{\text{selection}}$ remains at most $\mathcal{O}(nd_{\max}^2)$. The above cache scheme is special to our method and cannot be used with Smola and Bartlett's method without unduly increasing its complexity. If available, it is also useful to have an extra cache for storing kernel rows of instances which get discarded in one step, but which get to be considered again in a future step. Smola and Bartlett's method can also gain from such a cache.

## 4 Model adaptation

In this section we address the problem of model adaptation for a given number of basis functions, $d_{\max}$. Seeger (2003) and Seeger et al. (2003) give the details together with a very good discussion of various issues associated with gradient based model adaptation. Since the same ideas hold for all basis selection methods, we will not discuss them in detail. The sparse GP model is conditional on the parameters in the kernel function and the Gaussian noise level $\sigma^2$, which can all be collected together in $\theta$, the hyperparameter vector. The optimal values of $\theta$ can be inferred by minimizing the negative log

of the marginal likelihood, $\phi(\theta) = -\log \mathcal{P}(\boldsymbol{y}|\theta)$ using gradient based techniques, where $\mathcal{P}(\boldsymbol{y}|\theta) = \int \mathcal{P}(\boldsymbol{y}|\boldsymbol{f}_{\mathcal{I}})\mathcal{P}(\boldsymbol{f}_{\mathcal{I}})d\boldsymbol{f}_{\mathcal{I}} = \mathcal{N}(\boldsymbol{y}|0, \sigma^2\mathbf{I} + \mathbf{K}_{\mathcal{I},\cdot}^T\mathbf{K}_{\mathcal{I}}^{-1}\mathbf{K}_{\mathcal{I},\cdot})$. One of the problems in doing this is the dependence of $\mathcal{I}$ on $\theta$ that makes $\phi$ a non-differentiable function. This problem can be handled by repeating the following alternating steps: (1) fix $\theta$ and select $\mathcal{I}$ by the given basis selection algorithm; and (2) fix $\mathcal{I}$ and do a (short) gradient based adaptation of $\theta$. For the cache-based post-backfitting method of basis selection we also do the following for adding some stability to the model adaptation process. After we do step (2) using some $\mathcal{I}$ and obtain a $\theta$ we set the initial kernel cache, $\mathcal{C}$ using the rows of $\mathbf{K}_{\mathcal{I},\cdot}$ at $\theta$.

## 5   Numerical experiments

In this section, we compare our method against other sparse GP methods to verify the usefulness of our algorithm. To evaluate generalization performance, we utilize *Normalized Mean Square Error* (NMSE) given by $\frac{1}{t}\sum_{i=1}^{t}\frac{(y_i-\mu_i)^2}{\text{Var}(\boldsymbol{y})}$ and *Negative Logarithm of Predictive Distribution* (NLPD) defined as $\frac{1}{t}\sum_{i=1}^{t}-\log\mathcal{P}(y_i|\mu_i,\sigma_i^2)$ where $t$ is the number of test cases, $y_i$, $\mu_i$ and $\sigma_i^2$ are, respectively, the target, the predictive mean and the predictive variance of the $i$-th test case. NMSE uses only the mean while NLPD measures the quality of predictive distributions as it penalizes over-confident predictions as well as under-confident ones. For all experiments, we use the ARD Gaussian kernel defined by $\mathcal{K}(\boldsymbol{x}_i, \boldsymbol{x}_j) = v_0\exp\left(\sum_{\ell=1}^{m}v_\ell(\boldsymbol{x}_i^\ell - \boldsymbol{x}_j^\ell)^2\right) + v_b$ where $v_0, v_\ell, v_b > 0$ and $\boldsymbol{x}_i^\ell$ denotes the $\ell$-th element of $\boldsymbol{x}_i$. The ARD parameters $\{v_\ell\}$ give variable weights to input features that leads to a type of feature selection.

**Quality of Basis Selection in KIN40K Data Set.** We use the KIN40K data set,[1] composed of 40,000 samples, to evaluate and compare the performance of the various basis selection criteria. We first trained a full GPR model with the ARD Gaussian kernel on a subset of 2000 samples randomly selected in the dataset. The optimal values of the hyperparameters that we obtained were fixed and used for all the sparse GP models in this experiment. We compare the following five basis selection methods:

1. the baseline algorithm (RAND) that selects $\mathcal{I}$ at random;
2. the information gain algorithm (INFO) proposed by Seeger et al. (2003);
3. our algorithm described in Section 3 with cache size $c = \kappa = 59$ (KAPPA) in which we evaluate the selection scores of $\kappa$ instances at each step;
4. our algorithm described in Section 3 with cache size $c = d_{\max}$ (DMAX);
5. the algorithm (SB) proposed by Smola and Bartlett (2001) with $\kappa = 59$.

We randomly selected 10,000 samples for training, and kept the remaining 30,000 samples as test cases. For the purpose of studying variability the methods were run on ten such random partitions. We varied $d_{\max}$ from 100 to 1200. The test performances of the five methods are presented in Figure 1. From the upper plot of Figure 1 we can see that INFO yields much worse NMSE results than KAPPA, DMAX and SB, when $d_{\max}$ is less than 600. When the size is around 100, INFO is even worse than RAND. DMAX is always better than KAPPA. Interestingly, DMAX is even slightly better than SB when $d_{\max}$ is less than 200. This is probably because DMAX has a bigger set of basis functions to choose from, than SB. SB generally yields slightly better results than KAPPA. From the middle plot of Figure 1 we can note that INFO always gives poor NLPD results, even worse than RAND. The performances of KAPPA, DMAX and SB are close.

The lower plot of Figure 1 gives the CPU time consumed by the five algorithms for training, as a function of $d_{\max}$, in $\log - \log$ scale. The scaling exponents of RAND, INFO and SB are

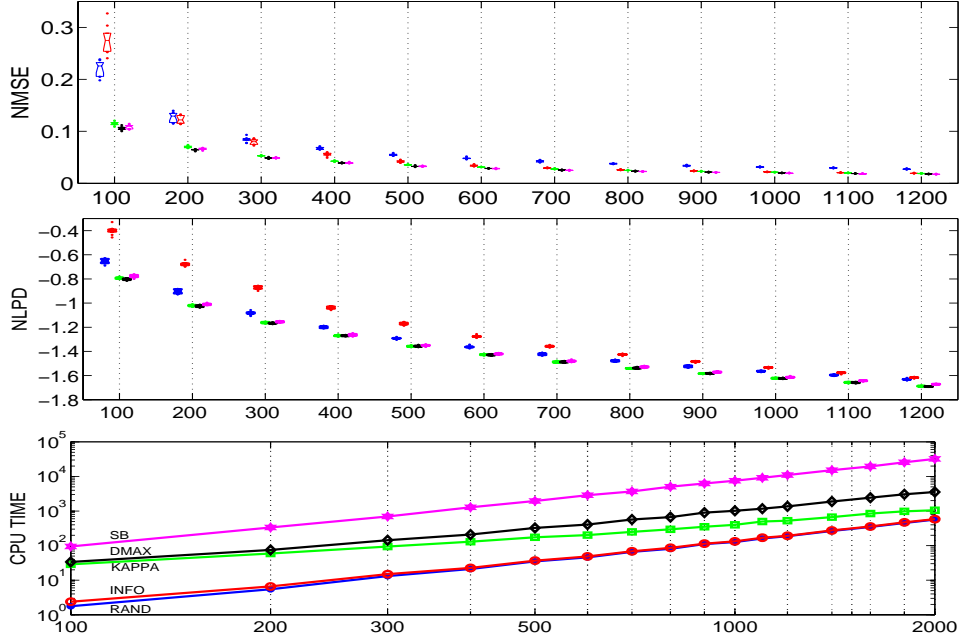

Figure 1: The variations of test set NMSE, test set NLPD and CPU time (in seconds) for training of the five algorithms as a function of $d_{\max}$. In the NMSE and NLPD plots, at each value of $d_{\max}$, the results of the five algorithms are presented as a boxplot group. From left to right, they are RAND(blue), INFO(red), KAPPA(green), DMAX(black), and SB(magenta). Note that the CPU time plot is on a $\log - \log$ scale.

around 2.0 (i.e., cost is proportional to $d_{\max}^2$), which is consistent with our analysis. INFO is almost as fast as RAND, while SB is about 60 times slower than INFO. The gap between KAPPA and INFO is the $\mathcal{O}(\kappa n d_{\max})$ time in computing the score (8) for $\kappa$ candidates.[2] As $d_{\max}$ increases, the cost of KAPPA asymptotically gets close to INFO. The gap between DMAX and KAPPA is the $\mathcal{O}(n d_{\max}^2 - \kappa n d_{\max})$ cost in computing the score (8) for the additional $(d_{\max} - \kappa)$ instances. Thus, as $d_{\max}$ increases, the curve of DMAX asymptotically becomes parallel to the curve of INFO. Asymptotically, the ratio of the computational times of DMAX and INFO is only about 3. Thus, unlike SB, which is about 60 times slower than INFO, DMAX is only about 3 times slower than INFO. Thus DMAX is an excellent method for achieving excellent generalization while also being quite efficient.

**Model Adaptation on Benchmark Data Sets.** Next, we compare model adaptation abilities of the following three algorithms for $d_{\max} = 500$.

1. The SB algorithm is applied to build a sparse GPR model with fixed hyperparameters (FIXED-SB). The values of these hyperparameters were obtained by training via a standard full GPR model on a manageable subset of 2000 samples randomly selected from the training data. FIXED-SB serves as a baseline.

2. The model adaptation scheme is coupled with the INFO basis selection algorithm (ADAPT-INFO).

3. The model adaptation scheme is coupled with our DMAX basis selection algorithm (ADAPT-DMAX).

Table 1: Test results of the three algorithms on the seven benchmark regression datasets. The results are the averages over 20 trials, along with the standard deviation. $d$ denotes the number of input features, $n_{trg}$ denotes the training data size and $n_{tst}$ denotes the test data size. We use bold face to indicate the lowest average value among the results of the three algorithms. The symbol $\star$ is used to indicate the cases significantly worse than the winning entry; a p-value threshold of 0.01 in Wilcoxon rank sum test was used to decide this.

| DATASET | $d$ | $n_{trg}$ | $n_{tst}$ | N M S E | | | N L P D | | |
|---|---|---|---|---|---|---|---|---|---|
| | | | | FIXED-SB | ADAPT-INFO | ADAPT-DMAX | FIXED-SB | ADAPT-INFO | ADAPT-DMAX |
| BANK8FM | 8 | 4500 | 3692 | **3.52 ± 0.08** | 3.54 ± 0.08 | 3.56 ± 0.09 | 3.11 ± 0.65$^\star$ | 1.37 ± 0.34$^\star$ | **0.67 ± 0.53** |
| BANK32NH | 32 | 4500 | 3692 | 48.08 ± 2.92 | 49.04 ± 1.34$^\star$ | **47.41 ± 1.35** | **−1.02 ± 0.21** | −0.79 ± 0.06$^\star$ | −0.88 ± 0.03$^\star$ |
| CPUSMALL | 12 | 4500 | 3692 | 2.45 ± 0.16 | **2.45 ± 0.15** | 2.46 ± 0.14 | 5.18 ± 0.61$^\star$ | 3.70 ± 0.46$^\star$ | **3.04 ± 0.17** |
| CPUACT | 21 | 4500 | 3692 | **1.58 ± 0.13** | 1.61 ± 0.14 | 1.61 ± 0.11 | 4.49 ± 0.26$^\star$ | 3.68 ± 0.40$^\star$ | **3.09 ± 0.20** |
| CALHOUSE | 8 | 10000 | 10640 | 22.58 ± 0.34$^\star$ | 22.82 ± 0.46$^\star$ | **20.02 ± 0.88** | 31.83 ± 3.35$^\star$ | 21.20 ± 1.47$^\star$ | **13.03 ± 0.30** |
| HOUSE8L | 8 | 10000 | 12784 | 42.27 ± 2.14$^\star$ | 37.30 ± 1.29$^\star$ | **35.87 ± 0.94** | 12.06 ± 0.67 | 12.06 ± 0.07$^\star$ | **11.71 ± 0.03** |
| HOUSE16H | 16 | 10000 | 12784 | 53.45 ± 7.05$^\star$ | 45.72 ± 1.15$^\star$ | **44.29 ± 0.76** | 12.72 ± 1.69 | 12.48 ± 0.06$^\star$ | **12.13 ± 0.04** |

We selected seven large regression datasets.[3] Each of them is randomly partitioned into training/test splits. For the purpose of analyzing statistical significance, the partition was repeated 20 times independently. Test set performances (NMSE and NLPD) of the three methods on the seven datasets are presented in Table 1. On the four datasets with 4500 training instances, the NMSE results of the three methods are quite comparable. ADAPT-DMAX yields significantly better NLPD results on three of those four datasets. On the three larger datasets with 10,000 training instances, ADAPT-DMAX is significantly better than ADAPT-INFO on both NMSE and NLPD.

We also tested our algorithm on the Outaouais dataset, which consists of 29000 training samples and 20000 test cases whose targets are held by the organizers of the "Evaluating Predictive Uncertainty Challenge".[4] The results of NMSE and NLPD we obtained in this blind test are $0.014$ and $-1.037$ respectively, which are much better than the results of other participants.

## Footnotes

[1]The dataset is available at http://www.igi.tugraz.at/aschwaig/data.html.

[2]If we want to take kernel evaluations also into account, the cost of KAPPA is $\mathcal{O}(m\kappa n d_{\max})$ where $m$ is the number of input variables. Note that INFO does not require any kernel evaluations for computing its selection criterion.

[3] These datasets are vailable at http://www.liacc.up.pt/~ltorgo/Regression/DataSets.html.

[4] The dataset and the results contributed by other participants can be found at the web site of the challenge http://predict.kyb.tuebingen.mpg.de/.

# References

Adler, J., B. D. Rao, and K. Kreutz-Delgado. Comparison of basis selection methods. In *Proceedings of the 30th Asilomar conference on signals, systems and computers*, pages 252–257, 1996.

Candela, J. Q. *Learning with uncertainty - Gaussian processes and relevance vector machines*. PhD thesis, Technical University of Denmark, 2004.

Csató, L. and M. Opper. Sparse online Gaussian processes. *Neural Computation, The MIT Press*, 14:641–668, 2002.

Seeger, M. *Bayesian Gaussian process models: PAC-Bayesian generalisation error bounds and sparse approximations*. PhD thesis, University of Edinburgh, July 2003.

Seeger, M., C. K. I. Williams, and N. Lawrence. Fast forward selection to speed up sparse Gaussian process regression. In *Workshop on AI and Statistics 9*, 2003.

Smola, A. J. and P. Bartlett. Sparse greedy Gaussian process regression. In Leen, T. K., T. G. Dietterich, and V. Tresp, editors, *Advances in Neural Information Processing Systems 13*, pages 619–625. MIT Press, 2001.

Vincent, P. and Y. Bengio. Kernel matching pursuit. *Machine Learning*, 48:165–187, 2002.

Williams, C. K. I. and M. Seeger. Using the Nyström method to speed up kernel machines. In Leen, T. K., T. G. Dietterich, and V. Tresp, editors, *Advances in Neural Information Processing Systems 13*, pages 682–688. MIT Press, 2001.

